# Discontinuous Generalization in Large Committee Machines

**H. Schwarze**
Dept. of Theoretical Physics
Lund University
Sölvegatan 14A
223 62 Lund
Sweden

**J. Hertz**
Nordita
Blegdamsvej 17
2100 Copenhagen Ø
Denmark

## Abstract

The problem of learning from examples in multilayer networks is studied within the framework of statistical mechanics. Using the replica formalism we calculate the average generalization error of a fully connected committee machine in the limit of a large number of hidden units. If the number of training examples is proportional to the number of inputs in the network, the generalization error as a function of the training set size approaches a finite value. If the number of training examples is proportional to the number of weights in the network we find first–order phase transitions with a discontinuous drop in the generalization error for both binary and continuous weights.

## 1   INTRODUCTION

Feedforward neural networks are widely used as nonlinear, parametric models for the solution of classification tasks and function approximation. Trained from examples of a given task, they are able to generalize, i.e. to compute the correct output for new, unknown inputs. Since the seminal work of Gardner (Gardner, 1988) much effort has been made to study the properties of feedforward networks within the framework of statistical mechanics; for reviews see e.g. (Hertz et al., 1989; Watkin et al., 1993). Most of this work has concentrated on the simplest feedforward network, the simple perceptron with only one layer of weights connecting the inputs with a

single output. However, most applications have to utilize architectures with hidden layers, for which only a few general theoretical results are known, e.g. (Levin et al., 1989; Krogh and Hertz, 1992; Seung et al., 1992).

As an example of a two–layer network we study the *committee machine* (Nilsson, 1965). This architecture has only one layer of adjustable weights, while the hidden–to–output weights are fixed to +1 so as to implement a majority decision of the hidden units. For binary weights this may already be regarded as the most general two–layer architecture, because any other combination of hidden–output weights can be gauged to +1 by flipping the signs of the corresponding input–hidden weights. Previous work has been concerned with some restricted versions of this model, such as learning geometrical tasks in machines with local input–to–hidden connectivity (Sompolinsky and Tishby, 1990) and learning in committee machines with nonoverlapping receptive fields (Schwarze and Hertz, 1992; Mato and Parga, 1992). In this tree–like architecture there are no correlations between hidden units and its behavior was found to be qualitatively similar to the simple perceptron.

Recently, learning in fully connected committee machines has been studied within the annealed approximation (Schwarze and Hertz, 1993a,b; Kang et al, 1993), revealing properties which are qualitatively different from the tree model. However, the annealed approximation (AA) is only valid at high temperatures, and a correct description of learning at low temperatures requires the solution of the quenched theory. The purpose of this paper is to extend previous work towards a better understanding of the learning properties of multilayer networks. We present results for the average generalization error of a fully connected committee machine within the replica formalism and compare them to results obtained within the AA. In particular we consider a large–net limit in which both the number of inputs $N$ and the number of hidden units $K$ go to infinity but with $K \ll N$. The target rule is defined by another fully connected committee machine and is therefore realizable by the learning network.

## 2   THE MODEL

We consider a network with $N$ inputs, $K$ hidden units and a single output unit $\sigma$. Each hidden unit $\sigma_l$, $l \in \{1, \ldots, K\}$, is connected to the inputs $\underline{S} = (S_1, \ldots, S_N)$ through the weight vector $\underline{W}_l$ and performs the mapping

$$\sigma_l(\underline{W}_l, \underline{S}) = \text{sign}\left(\frac{1}{\sqrt{N}} \underline{W}_l \cdot \underline{S}\right). \tag{1}$$

The hidden units may be regarded as outputs of simple perceptrons and will be referred to as *students*. The factor $N^{-1/2}$ in (1) is included for convenience; it ensures that in the limit $N \to \infty$ and for *iid* inputs the argument of the sign function is of order 1. The overall network output is defined as the majority vote of the student committee, given by

$$\sigma(\{\underline{W}_l\}, \underline{S}) = \text{sign}\left(\frac{1}{\sqrt{K}} \sum_{l=1}^{K} \sigma_l(\underline{W}_l, \underline{S})\right). \tag{2}$$

This network is trained from $P = \alpha KN$ input–output examples $(\underline{\xi}^\mu, \tau(\underline{\xi}^\mu))$, $\mu \in \{1, \ldots, P\}$, of the desired mapping $\tau$, where the components $\xi_i^\mu$ of the training inputs are independently drawn from a distribution with zero mean and unit variance. We study a realizable task defined by another committee machine with weight vectors $\underline{V}_l$ (the *teachers*), hidden units $\tau_l$ and an overall output $\tau(\underline{S})$ of the form (2). We will discuss both the binary version of this model with $\underline{W}_l, \underline{V}_l \in \{\pm 1\}^N$ and the continuous version in which the $\underline{W}_l$'s and $\underline{V}_l$'s are normalized to $\sqrt{N}$.

The goal of learning is to find a network that performs well on unknown examples, which are not included in the training set. The network quality can be measured by the generalization error

$$\epsilon(\{\underline{W}_l\}) = \langle\, \Theta[-\sigma(\{\underline{W}_l\}, \underline{S})\, \tau(\underline{S})]\,\rangle_{\underline{S}}, \tag{3}$$

the probability that a randomly chosen input is misclassified.

Following the statistical mechanics approach we consider a stochastic learning algorithm that for long training times yields a Gibbs distribution of networks with the corresponding partition function

$$Z = \int d\rho_0(\{\underline{W}_l\})\, e^{-\beta E_t(\{\underline{W}_l\})}, \tag{4}$$

where

$$E_t(\{\underline{W}_l\}) = \sum_\mu \Theta\big[-\sigma(\{\underline{W}_l\}, \underline{\xi}^\mu)\, \tau(\underline{\xi}^\mu)\big] \tag{5}$$

is the training error, $\beta = 1/T$ is a formal temperature parameter, and $\rho_0(\{\underline{W}_l\})$ includes *a priori* constraints on the weights. The average generalization and training errors at thermal equilibrium, averaged over all representations of the training examples, are given by

$$\begin{aligned} \epsilon_g &= \langle\!\langle\,\langle\, \epsilon(\{\underline{W}_l\})\,\rangle_T\,\rangle\!\rangle \\ \epsilon_t &= \frac{1}{P}\, \langle\!\langle\,\langle\, E_t(\{\underline{W}_l\})\,\rangle_T\,\rangle\!\rangle, \end{aligned} \tag{6}$$

where $\langle\!\langle \ldots \rangle\!\rangle$ denotes a quenched average over the training examples and $\langle \ldots \rangle_T$ a thermal average. These quantities may be obtained from the average free energy $F = -T\langle\!\langle \ln Z \rangle\!\rangle$, which can be calculated within the standard replica formalism (Gardner, 1988; Györgyi and Tishby, 1990).

Following this approach, we introduce order parameters and make symmetry assumptions for their values at the saddle point of the free energy; for details of the calculation see (Schwarze, 1993). We assume replica symmetry (RS) and a partial committee symmetry allowing for a specialization of the hidden units on their respective teachers. Furthermore, a self–consistent solution of the saddle–point equations requires scaling assumptions for the order parameters. Hence, we are left with the ansatz

$$\begin{aligned} R_{lk} &= \frac{1}{N}\langle\!\langle\,\langle\,\underline{W}_l\,\rangle_T \cdot \underline{V}_k\,\rangle\!\rangle &&= \frac{\rho}{K} + \Delta\,\delta_{lk} \\ D_{lk} &= \frac{1}{N}\langle\!\langle\,\langle\,\underline{W}_l\,\rangle_T \cdot \langle\!\langle\,\langle\,\underline{W}_k\,\rangle_T\,\rangle\!\rangle &&= \frac{d}{K} + q\,\delta_{lk} \\ C_{lk} &= \frac{1}{N}\langle\!\langle\,\langle\,\underline{W}_l \cdot \underline{W}_k\,\rangle_T\,\rangle\!\rangle &&= \frac{c}{K} + (1 - \frac{c}{K})\,\delta_{lk}, \end{aligned} \tag{7}$$

where $\rho$, $\Delta$, $d$, $q$ and $c$ are of order 1. For $\Delta = q = 0$ this solution is symmetric under permutations of hidden units in the student network, while nonvanishing $\Delta$ and $q$ indicate a specialization of hidden units that breaks this symmetry. The values of the order parameters at the saddle point of the replica free energy finally allow the calculation of the average generalization and training errors.

## 3  THEORETICAL RESULTS

In the limit of small training set sizes, $\alpha \sim \mathcal{O}(1/K)$, we find a committee–symmetric solution where each student weight vector has the same overlap to all the teacher vectors, corresponding to $\Delta = q = 0$. For both binary and continuous weights the generalization error of this solution approaches a nonvanishing residual value as shown in figure 1. Note that the asymptotic generalization ability of the committee–symmetric solution improves with increasing noise level.

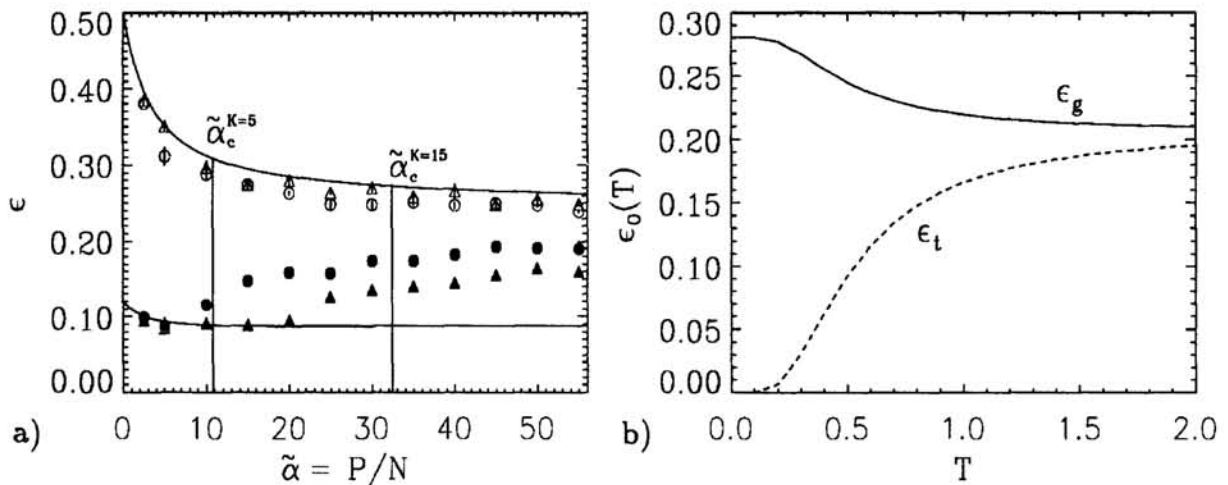

Figure 1: a) Generalization (upper curve) and training (lower curve) error as functions of $\tilde{\alpha} = K\alpha$. The results of Monte Carlo simulations for the generalization (open symbols) and training (closed symbols) errors are shown for $K = 5$ (circles) and $K = 15$ (triangles) with $T = 0.5$ and $N = 99$. The vertical lines indicate the predictions of the large–$K$ theory for the location of the phase transition $\tilde{\alpha}_c = K\alpha_c$ in the binary model for $K = 5$ and $K = 15$, respectively.
b) Temperature dependence of the asymptotic generalization and training errors for the committee–symmetric solution.

Only if the number of training examples is sufficiently large, $\alpha \sim \mathcal{O}(1)$, can the committee symmetry be broken in favor of a specialization of hidden units. We find first–order phase transitions to solutions with $\Delta, q > 0$ in both the continuous and the binary model. While in the binary model the transition is accompanied by a perfect alignment of the hidden–unit weight vectors with their respective teachers ($\Delta = 1$), this is not possible in a continuous model. Instead, we find a close approach of each student vector to one of the teachers in the continuous model: At a critical value $\alpha_s(T)$ of the load parameter a second minimum of the free energy appears, corresponding to the specialized solution with $\Delta, q > 0$. This solution becomes the

global minimum at $\alpha_c(T) > \alpha_s(T)$, and its generalization error decays algebraically. In both models the symmetric, poorly generalizing state remains metastable for arbitrarily large $\alpha$. For increasing system sizes it will take exponentially long times for a stochastic training algorithm to escape from this local minimum (see figure 1a). Figure 2 shows the qualitative behavior of the generalization error for the continuous model, and the phase diagrams in figure 3 show the location of the transitions for both models.

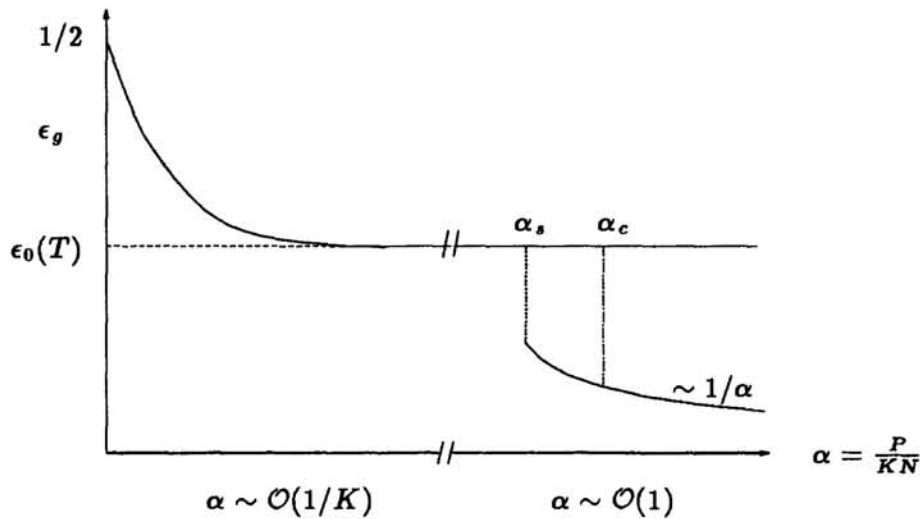

Figure 2: Schematic behavior of the generalization error in the large-$K$ committee machine with continuous weights.

In the binary model a region of negative thermodynamic entropy (below the dashed line in figure 3a) suggests that replica symmetry has to be broken to correctly describe the metastable, symmetric solution at large $\alpha$.

A comparison of the RS solution with the results previously obtained within the AA (Schwarze and Hertz, 1993a,b) shows that the AA gives a qualitatively correct description of the main features of the learning curve. However, it fails to predict the temperature dependence of the residual generalization error (figure 1b) and gives an incorrect description of the approach to this value. Furthermore, the quantitative predictions for the locations of the phase transitions differ considerably (figure 3).

## 4    SIMULATIONS

We have performed Monte Carlo simulations to check our analytical findings for the binary model (see figure 1a). The influence of the metastable, poorly generalizing state is reflected by the fact that at low temperatures the simulations do not follow the predicted phase transition but get trapped in the metastable state. Only at higher temperatures do the simulations follow the first order transition (Schwarze, 1993). Furthermore, the deviation of the training error from the theoretical result indicates the existence of replica symmetry breaking for finite $\alpha$. However, the generalization error of the symmetric state is in good quantitative agreement with the

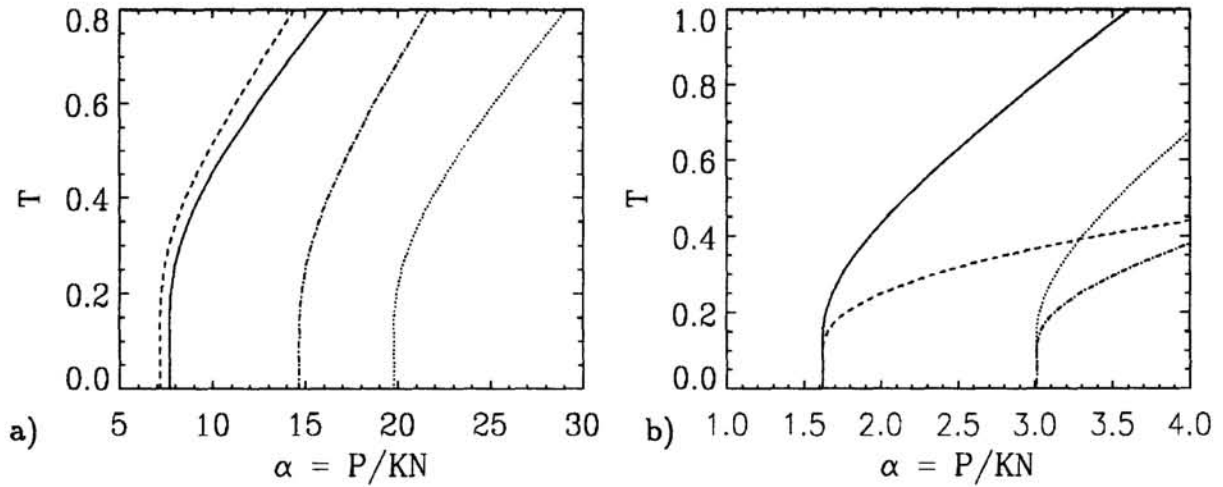

**Figure 3:** Phase diagrams of the large-$K$ committee machine.
a) continuous weights: The two left lines show the RS results for the spinodal line $(--)$, where the specialized solution appears, and the location of the phase transition $(\text{—})$. These results are compared to the predictions of the AA for the spinodal line $(-\cdot-)$ and the phase transition $(\cdots)$.
b) binary weights: The RS result for the location of the phase transition $(\text{—})$ and its zero–entropy line $(--)$ are compared to the prediction of the AA for the phase transition $(\cdots)$ and its zero–entropy line $(-\cdot-)$.

theoretical results.

In order to investigate whether our analytical results for a Gibbs ensemble of committee machines carries over to other learning scenarios we have studied a variation of this model allowing the use of backpropagation. We have considered a 'soft–committee' whose output is given by

$$\sigma(\{\underline{W}_l\}, \underline{S}) = \tanh\left(\sum_{l=1}^{K} \tanh\left(\underline{W}_l \cdot \underline{S}\right)\right). \tag{8}$$

The first–layer weights $\underline{W}_l$ of this network were trained on examples $(\underline{\xi}^\mu, \tau(\underline{\xi}^\mu))$, $\mu \in \{1, \ldots, P\}$, defined by another soft–committee with weight vectors $\underline{V}_l$ using on–line backpropagation with the error function

$$\epsilon(\underline{S}) = (1/2)[\sigma(\{\underline{W}_l\}, \underline{S}) - \tau(\underline{S})]^2. \tag{9}$$

In general this procedure is not guaranteed to yield a Gibbs distribution of weights (Hansen et al., 1993) and therefore the above analysis does not apply to this case. However, the generalization error for a network with $N = 45$ inputs and $K = 3$ hidden units, averaged over 50 independent runs, shows the same qualitative behavior as predicted for the Gibbs ensemble of committee machines (see figure 4). After an initial approach to a nonvanishing value, the average generalization error decreases rather smoothly to zero. This smooth decrease of the average error is due to the fact that some runs got trapped in a poorly–generalizing, committee–symmetric solution while others found a specialized solution with a close approach to the teacher.

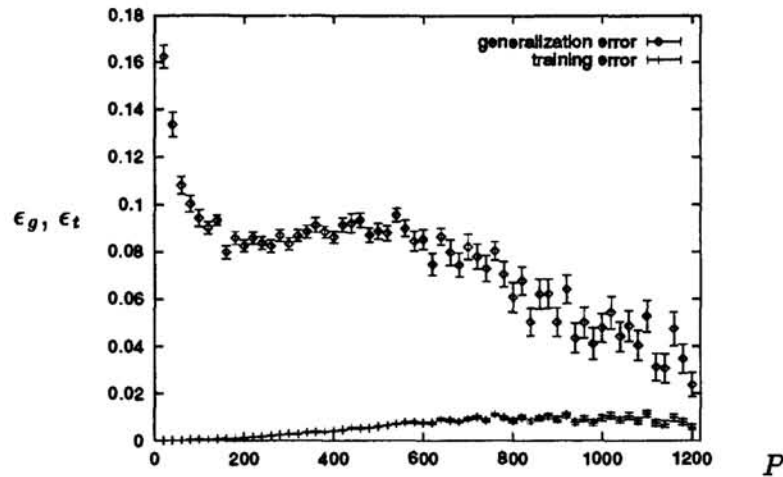

Figure 4: Generalization error and training error of the 'soft–committee' with $N = 45$ and $K = 3$. We have used standard on–line backpropagation for the first–layer weights with a learning rate $\eta = 0.01$ for 1000 epochs. the results are averaged over 50 runs with different teacher networks and different training sets.

## 5    CONCLUSION

We have presented the results of a calculation of the generalization error of a multi-layer network within the statistical mechanics approach. We have found nontrivial behavior for networks with both continuous and binary weights. In both models, phase transitions from a symmetric, poorly–generalizing solution to one with specialized hidden units occur, accompanied by a discontinuous drop of the generalization error. However, the existence of a metastable, poorly generalizing solution beyond the phase transition implies the possibility of getting trapped in a local minimum during the training process. Although these results were obtained for a Gibbs distribution of networks, numerical experiments indicate that some of the general results carry over to other learning scenarios.

### Acknowledgements

The authors would like to thank M. Biehl and S. Solla for fruitful discussions. HS acknowledges support from the EC under the SCIENCE programme (under grant number B/SC1*/915125) and by the Danish Natural Science Council and the Danish Technical Research Council through CONNECT.

### References

E. Gardner (1988), J. Phys. A **21**, 257.

G. Györgyi and N. Tishby (1990), in *Neural Networks and Spin Glasses*, edited by K. Thuemann and R. Köberle, (World scientific, Singapore).

L.K. Hansen, R. Pathria, and P. Salamon (1993), J. Phys. A **26**, 63.

J. Hertz, A. Krogh, and R.G. Palmer (1989), *Introduction to the Theory of Neural*

*Computation*, (Addison–Wesley, Redwood City, CA).

K. Kang, J.–H. Oh, C. Kwon, and Y. Park (1993), preprint Pohang Institute of Science and Technology, Korea.

A. Krogh and J. Hertz (1992), in *Advances in Neural Information Processing Systems IV*, eds. J.E. Moody, S.J. Hanson, and R.P. Lippmann, (Morgan Kaufmann, San Mateo).

E. Levin, N. Tishby, and S.A. Solla (1989), in *Proc. 2nd Workshop on Computational Learning Theory*, (Morgan Kaufmann, San Mateo).

G. Mato and N. Parga (1992), J. Phys. A **25**, 5047.

N.J. Nilsson (1965), *Learning Machines*, (McGraw–Hill, New York).

H. Schwarze (1993), J. Phys. A **26**, 5781.

H. Schwarze and J. Hertz (1992), Europhys. Lett. **20**, 375.

H. Schwarze and J. Hertz (1993a), J. Phys. A **26**, 4919.

H. Schwarze and J. Hertz (1993b), in *Advances in Neural Information Processing Systems V*, (Morgan Kaufmann, San Mateo).

H.S. Seung, H. Sompolinsky, and N. Tishby (1992), Phys. Rev. A **45**, 6056.

H. Sompolinsky and N. Tishby (1990), Europhys. Lett. **13**, 567.

T. Watkin, A. Rau, and M. Biehl (1993), Rev. Mod. Phys. **65**, 499.
